# The Nonnegative Boltzmann Machine

**Oliver B. Downs**
Hopfield Group
Schultz Building
Princeton University
Princeton, NJ 08544
obdowns@princeton.edu

**David J.C. MacKay**
Cavendish Laboratory
Madingley Road
Cambridge, CB3 0HE
United Kingdom
mackay@mrao.cam.ac.uk

**Daniel D. Lee**
Bell Laboratories
Lucent Technologies
700 Mountain Ave.
Murray Hill, NJ 07974
ddlee@bell-labs.com

## Abstract

The nonnegative Boltzmann machine (NNBM) is a recurrent neural network model that can describe multimodal nonnegative data. Application of maximum likelihood estimation to this model gives a learning rule that is analogous to the binary Boltzmann machine. We examine the utility of the mean field approximation for the NNBM, and describe how Monte Carlo sampling techniques can be used to learn its parameters. Reflective slice sampling is particularly well-suited for this distribution, and can efficiently be implemented to sample the distribution. We illustrate learning of the NNBM on a translationally invariant distribution, as well as on a generative model for images of human faces.

## Introduction

The multivariate Gaussian is the most elementary distribution used to model generic data. It represents the *maximum entropy* distribution under the constraint that the mean and covariance matrix of the distribution match that of the data. For the case of binary data, the maximum entropy distribution that matches the first and second order statistics of the data is given by the Boltzmann machine [1]. The probability of a particular state in the Boltzmann machine is given by the exponential form:

$$P(\{s_i = \pm 1\}) = \frac{1}{Z} \exp\left(-\frac{1}{2}\sum_{ij} s_i A_{ij} s_j + \sum_i b_i s_i\right). \tag{1}$$

Interpreting Eq. 1 as a neural network, the parameters $A_{ij}$ represent symmetric, recurrent weights between the different units in the network, and $b_i$ represent local biases. Unfortunately, these parameters are not simply related to the observed mean and covariance of the

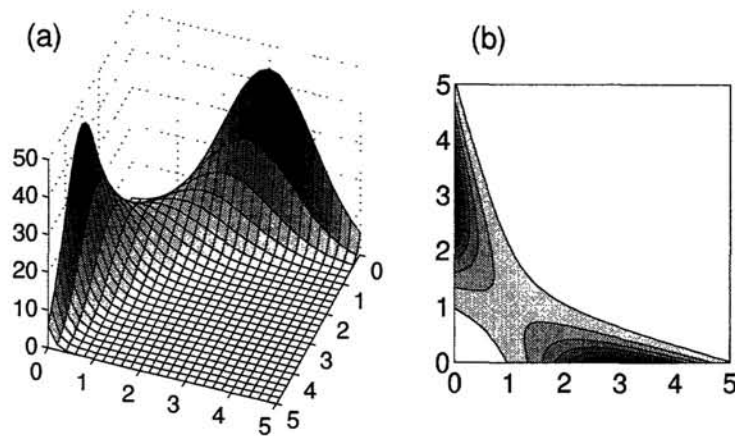

Figure 1: a) Probability density and b) shaded contour plot of a two dimensional competitive NNBM distribution. The energy function $E(x)$ for this distribution contains a saddle point and two local minima, which generates the observed multimodal distribution.

data as they are for the normal Gaussian. Instead, they need to be adapted using an iterative learning rule that involves difficult sampling from the binary distribution [2].

The Boltzmann machine can also be generalized to continuous and nonnegative variables. In this case, the maximum entropy distribution for nonnegative data with known first and second order statistics is described by a distribution previously called the "rectified Gaussian" distribution [3]:

$$P(x) = \begin{cases} \frac{1}{Z} \exp\left[-E(x)\right] & \text{if } x_i \geq 0 \ \forall i, \\ 0 & \text{if any } x_i < 0, \end{cases} \tag{2}$$

where the energy function $E(x)$ and normalization constant $Z$ are:

$$E(x) = \frac{1}{2} x^T A x - b^T x, \tag{3}$$

$$Z = \int_{x \geq 0} dx \ \exp[-E(x)]. \tag{4}$$

The properties of this nonnegative Boltzmann machine (NNBM) distribution differ quite substantially from that of the normal Gaussian. In particular, the presence of the nonnegativity constraints allows the distribution to have multiple modes. For example, Fig. 1 shows a two-dimensional NNBM distribution with two separate maxima located against the rectifying axes. Such a multimodal distribution would be poorly modelled by a single normal Gaussian.

In this submission, we discuss how a multimodal NNBM distribution can be learned from nonnegative data. We show the limitations of mean field approximations for this distribution, and illustrate how recent developments in efficient sampling techniques for continuous belief networks can be used to tune the weights of the network [4]. Specific examples of learning are demonstrated on a translationally invariant distribution, as well as on a generative model for face images.

## Maximum Likelihood

The learning rule for the NNBM can be derived by maximizing the log likelihood of the observed data under Eq. 2. Given a set of nonnegative vectors $\{\vec{x}^\mu\}$, where $\mu = 1..M$

indexes the different examples, the log likelihood is:

$$L = \frac{1}{M}\sum_{\mu=1}^{M}\log P(\vec{x}^{\mu}) = -\frac{1}{M}\sum_{\mu=1}^{M}E(\vec{x}^{\mu}) - \log Z. \tag{5}$$

Taking the derivatives of Eq. 5 with respect to the parameters $A$ and $b$ gives:

$$\frac{\partial L}{\partial A_{ij}} = \langle x_i x_j \rangle_{\mathrm{f}} - \langle x_i x_j \rangle_{\mathrm{c}} \tag{6}$$

$$\frac{\partial L}{\partial b_i} = \langle x_i \rangle_{\mathrm{c}} - \langle x_i \rangle_{\mathrm{f}}, \tag{7}$$

where the subscript "c" denotes a "clamped" average over the data, and the subscript "f" denotes a "free" average over the NNBM distribution:

$$\langle f(x) \rangle_{\mathrm{c}} = \frac{1}{M}\sum_{\mu=1}^{M}f(\vec{x}^{\mu}) \tag{8}$$

$$\langle f(x) \rangle_{\mathrm{f}} = \int_{x\geq 0} dx\, P(x)f(x). \tag{9}$$

These derivatives are used to define a gradient ascent learning rule for the NNBM that is similar to that of the binary Boltzmann machine. The contrast between the clamped and free covariance matrix is used to update the iterations $A$, while the difference between the clamped and free means is used to update the local biases $b$.

## Mean field approximation

The major difficulty with this learning algorithm lies in evaluating the averages $\langle x_i x_j \rangle_{\mathrm{f}}$ and $\langle x_i \rangle_{\mathrm{f}}$. Because it is analytically intractable to calculate these free averages exactly, approximations are necessary for learning. Mean field approximations have previously been proposed as a deterministic alternative for learning in the binary Boltzmann machine, although there have been contrasting views on their validity [5, 6]. Here, we investigate the utility of mean field theory for approximating the NNBM distribution.

The mean field equations are derived by approximating the NNBM distribution in Eq. 2 with the factorized form:

$$Q(x) = \prod_i Q_{\tau_i}(x_i) = \prod_i \frac{1}{\gamma!}\frac{1}{\tau_i}\left(\frac{x_i}{\tau_i}\right)^{\gamma}e^{-\frac{x_i}{\tau_i}}, \tag{10}$$

where the different marginal densities $Q(x_i)$ are characterized by the means $\tau_i$ with a fixed constant $\gamma$. The product of $\gamma$-distributions is the natural factorizable distribution for non-negative random variables.

The optimal mean field parameters $\tau_i$ are determined by minimizing the Kullback-Leibler divergence between the NNBM distribution and the factorized distribution:

$$D_{KL}(Q\|P) = \int dx\, Q(x)\log\left[\frac{Q(x)}{P(x)}\right] = \langle E(x)\rangle_{Q(x)} + \log Z - H(Q). \tag{11}$$

Finding the minimum of Eq. 11 by setting its derivatives with respect to the mean field parameters $\tau_i$ to zero gives the simple mean field equations:

$$A_{ii}\tau_i = (\gamma+1)\left[b_i - \sum_j A_{ij}\tau_j + \frac{1}{\tau_i}\right] \tag{12}$$

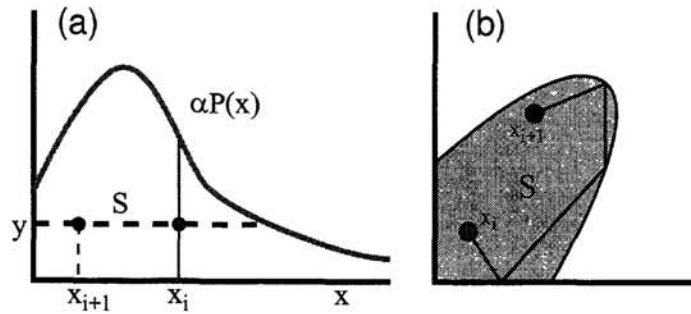

Figure 2: a) Slice sampling in one dimension. Given the current sample point, $x_i$, a height $y \in [0, \alpha P(x)]$ is randomly chosen. This defines a slice $(x \in S | \alpha P(x) \geq y)$ in which a new $x_{i+1}$ is chosen. b) For a multidimensional slice S, the new point $x_{i+1}$ is chosen using ballistic dynamics with specular reflections off the interior boundaries of the slice.

These equations can then be solved self-consistently for $\tau_i$. The "free" statistics of the NNBM are then replaced by their statistics under the factorized distribution $Q(x)$:

$$\langle x_i \rangle_{\mathrm{f}} \approx \tau_i, \quad \langle x_i x_j \rangle_{\mathrm{f}} \approx \left[ (\gamma + 1)^2 + (\gamma + 1)\, \delta_{ij} \right] \tau_i \tau_j. \tag{13}$$

The fidelity of this approximation is determined by how well the factorized distribution $Q(x)$ models the NNBM distribution. Unfortunately, for distributions such as the one shown in Fig. 3, the mean field approximation is quite different from that of the true multimodal NNBM distribution. This suggests that the naive mean field approximation is inadequate for learning in the NNBM, and in fact attempts to use this approximation fail to learn the examples given in following sections. However, the mean field approximation can still be used to initialize the parameters to reasonable values before using the sampling techniques that are described below.

## Monte-Carlo sampling

A more direct approach to calculating the "free" averages in Eq. 6–7 is to numerically approximate them. This can be accomplished by using Monte Carlo sampling to generate a representative set of points that sufficiently approximate the statistics of the continuous distribution. In particular, Markov chain Monte-Carlo methods employ an iterative stochastic dynamics whose equilibrium distribution converges to that of the desired distribution [4]. For the binary Boltzmann machine, such sampling dynamics involves random "spin flips" which change the value of a single binary component. Unfortunately, these single component dynamics are easily caught in local energy minima, and can converge very slowly for large systems. This makes sampling the binary distribution very difficult, and more specialized computational techniques such as simulated annealing, cluster updates, etc., have been developed to try to circumvent this problem.

For the NNBM, the use of continuous variables makes it possible to investigate different stochastic dynamics in order to more efficiently sample the distribution. We first experimented with Gibbs sampling with ordered overrelaxation [7], but found that the required inversion of the error function was too computationally expensive. Instead, the recently developed method of slice sampling [8] seems particularly well-suited for implementation in the NNBM.

The basic idea of the slice sampling algorithm is shown in Fig. 2. Given a sample point $x_i$, a random $y \in [0, \alpha P(x_i)]$ is first uniformly chosen. Then a slice $S$ is defined as the connected set of points $(x \in S \mid \alpha P(x) \geq y)$, and the new point $x_{i+1} \in S$ is chosen

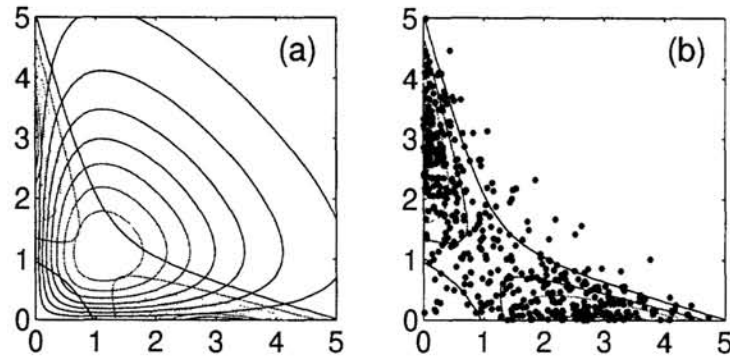

Figure 3: Contours of the two-dimensional competitive NNBM distribution overlaid by a) $\gamma = 1$ mean field approximation and b) 500 reflected slice samples.

randomly from this slice. The distribution of $x_n$ for large $n$ can be shown to converge to the desired density $P(x)$. Now, for the NNBM, solving the boundary points along a particular direction in a given slice is quite simple, since it only involves solving the roots of a quadratic equation. In order to efficiently choose a new point within a particular slice, reflective "billiard ball" dynamics are used. A random initial velocity is chosen, and the new point is evolved by travelling a certain distance from the current point while specularly reflecting from the boundaries of the slice. Intuitively, the reversibility of these reflections allows the dynamics to satisfy detailed balance.

In Fig. 3, the mean field approximation and reflective slice sampling are used to model the two-dimensional competitive NNBM distribution. The poor fit of the mean field approximation is apparent from the unimodality of the factorized density, while the sample points from the reflective slice sampling algorithm are more representative of the underlying NNBM distribution. For higher dimensional data, the mean field approximation becomes progressively worse. It is therefore necessary to implement the numerical slice sampling algorithm in order to accurately approximate the NNBM distribution.

## Translationally invariant model

Ben-Yishai et al. have proposed a model for orientation tuning in primary visual cortex that can be interpreted as a cooperative NNBM distribution [9]. In the absence of visual input, the firing rates of $N$ cortical neurons are described as minimizing the energy function $E(x)$ with parameters:

$$\frac{A_{ij}}{\beta} = \delta_{ij} + \frac{1}{N} - \frac{\epsilon}{N}\cos(\frac{2\pi}{N}|i-j|) \tag{14}$$

$$\frac{b_i}{\beta} = 1$$

This distribution was used to test the NNBM learning algorithm. First, a large set of $N = 25$ dimensional nonnegative training vectors were generated by sampling the distribution with $\beta = 50$ and $\epsilon = 4$. Using these samples as training data, the $A$ and $b$ parameters were learned from a unimodal initialization by evolving the training vectors using reflective slice sampling, and these evolved vectors were used to calculate the "free" averages in Eq. 6–7. The $A$ and $b$ estimates were then updated, and this procedure was iterated until the evolved averages matched that of the training data. The learned $A$ and $b$ parameters were then found to almost exactly match the original form in Eq. 14. Some representative samples from the learned NNBM distribution are shown in Fig. 4.

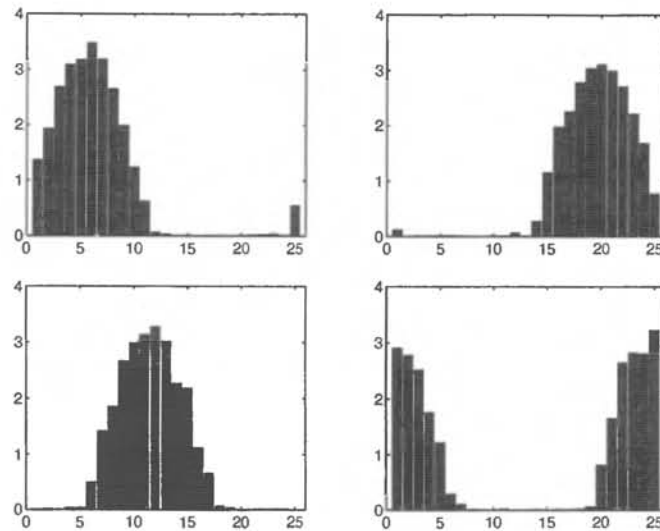

Figure 4: Representative samples taken from a NNBM after training to learn a translationally invariant cooperative distribution with $\beta = 50$ and $\epsilon = 4$.

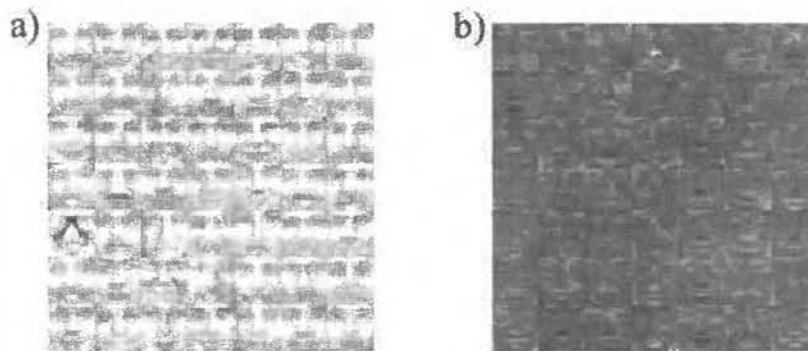

Figure 5: a) Morphing of a face image by successive sampling from the learned NNBM distribution. b) Samples generated from a normal Gaussian.

## Generative model for faces

We have also used the NNBM to learn a generative model for images of human faces. The NNBM is used to model the correlations in the coefficients of the nonnegative matrix factorization (NMF) of the face images [10]. NMF reduces the dimensionality of nonnegative data by decomposing the face images into parts correponding to eyes, noses, ears, etc. Since the different parts are coactivated in reconstructing a face, the activations of these parts contain significant correlations that need to be captured by a generative model. Here we briefly demonstrate how the NNBM is able to learn these correlations.

Sampling from the NNBM stochastically generates coefficients which can graphically be displayed as face images. Fig. 5 shows some representative face images as the reflective slice sampling dynamics evolves the coefficients. Also displayed in the figure are the analogous images generated if a normal Gaussian is used to model the correlations instead. It is clear that the nonnegativity constraints and multimodal nature of the NNBM results in samples which are cleaner and more distinct as faces.

## Discussion

Here we have introduced the NNBM as a recurrent neural network model that is able to describe multimodal nonnegative data. Its application is made practical by the efficiency of the slice sampling Monte Carlo method. The learning algorithm incorporates numerical sampling from the NNBM distribution and is able to learn from observations of nonnegative data. We have demonstrated the application of NNBM learning to a cooperative, translationally invariant distribution, as well as to real data from images of human faces.

Extensions to the present work include incorporating hidden units into the recurrent network. The addition of hidden units implies modelling certain higher order statistics in the data, and requires calculating averages over these hidden units. We anticipate the marginal distribution over these units to be most commonly unimodal, and hence mean field theory should be valid for approximating these averages.

Another possible extension involves generalizing the NNBM to model continuous data confined within a certain range, i.e. $0 \leq x_i \leq 1$. In this situation, slice sampling techniques would also be used to efficiently generate representative samples. In any case, we hope that this work stimulates more research into using these types of recurrent neural networks to model complex, multimodal data.

## Acknowledgements

The authors acknowledge useful discussion with John Hopfield, Sebastian Seung, Nicholas Socci, and Gayle Wittenberg, and are indebted to Haim Sompolinsky for pointing out the maximum entropy interpretation of the Boltzmann machine. This work was funded by Bell Laboratories, Lucent Technologies.

O.B. Downs is grateful for the moral support, and open ears and minds of Beth Brittle, Gunther Lenz, and Sandra Scheitz.

## References

[1] Hinton, GE & Sejnowski, TJ (1983). Optimal perceptual learning. *IEEE Conference on Computer Vision and Pattern Recognition*, Washington, DC, 448–453.

[2] Ackley, DH, Hinton, GE, & Sejnowski, TJ (1985). A learning algorithm for Boltzmann machines. *Cognitive Science* **9**, 147–169.

[3] Socci, ND, Lee, DD, and Seung, HS (1998). The rectified Gaussian distribution. *Advances in Neural Information Processing Systems* **10**, 350–356.

[4] MacKay, DJC (1998). Introduction to Monte Carlo Methods. *Learning in Graphical Models.* Kluwer Academic Press, NATO Science Series, 175–204.

[5] Galland, CC (1993). The limitations of deterministic Boltzmann machine learning. *Network* **4**, 355–380.

[6] Kappen, HJ & Rodriguez, FB (1997). Mean field approach to learning in Boltzmann machines. *Pattern Recognition in Practice V)*, Amsterdam.

[7] Neal, RM (1995). Suppressing random walks in Markov chain Monte Carlo using ordered overrelaxation. Technical Report 9508, Dept. of Statistics, University of Toronto.

[8] Neal, RM (1997). Markov chain Monte Carlo methods based on "slicing" the density function. Technical Report 9722, Dept. of Statistics, University of Toronto.

[9] Ben-Yishai, R, Bar-Or, RL, & Sompolinsky, H (1995). Theory of orientation tuning in visual cortex. *Proc. Nat. Acad. Sci. USA* **92**, 3844–3848.

[10] Lee, DD, and Seung, HS (1999) Learning the parts of objects by non-negative matrix factorization. *Nature* **401**, 788-791.